# Shifting Weights: Adapting Object Detectors from Image to Video

**Kevin Tang**[1]     **Vignesh Ramanathan**[2]     **Li Fei-Fei**[1]     **Daphne Koller**[1]

[1]Computer Science Department, Stanford University, Stanford, CA 94305
[2]Department of Electrical Engineering, Stanford University, Stanford, CA 94305
{kdtang,vigneshr,feifeili,koller}@cs.stanford.edu

## Abstract

Typical object detectors trained on images perform poorly on video, as there is a clear distinction in domain between the two types of data. In this paper, we tackle the problem of adapting object detectors learned from images to work well on videos. We treat the problem as one of unsupervised domain adaptation, in which we are given labeled data from the source domain (image), but only unlabeled data from the target domain (video). Our approach, self-paced domain adaptation, seeks to iteratively adapt the detector by re-training the detector with automatically discovered target domain examples, starting with the easiest first. At each iteration, the algorithm adapts by considering an increased number of target domain examples, and a decreased number of source domain examples. To discover target domain examples from the vast amount of video data, we introduce a simple, robust approach that scores trajectory tracks instead of bounding boxes. We also show how rich and expressive features specific to the target domain can be incorporated under the same framework. We show promising results on the 2011 TRECVID Multimedia Event Detection [1] and LabelMe Video [2] datasets that illustrate the benefit of our approach to adapt object detectors to video.

## 1   Introduction

Following recent advances in learning algorithms and robust feature representations, tasks in video understanding have shifted from classifying simple motions and actions [3, 4] to detecting complex events and activities in Internet videos [1,5,6]. Detecting complex events is a difficult task, requiring probabilistic models that can understand the semantics of what is occuring in the video. Because many events are characterized by key objects and their interactions, it is imperative to have robust object detectors that can provide accurate detections. In this paper, we focus on the problem of detecting objects in complex Internet videos. It is difficult to obtain labeled objects in these types of videos because of the large number of frames, and the fact that objects may not appear in many of them. Thus, a common approach is to train object detectors from labeled images, which are widely available. However, as seen in Figure 1, the domain of images and videos is quite different, as it is often the case that images of objects are taken in controlled settings that differ greatly from where they appear in real-world situations, as seen in video. Thus, we cannot typically expect a detector trained on images to work well in videos.

To adapt object detectors from image to video, we take an incremental, self-paced approach to learn from the large amounts of unlabeled video data available. We make the assumption that within our unlabeled video data, there exist instances of our target object. However, we do not assume that every video has an instance of the object, due to the noise present in Internet videos. We start by introducing a simple, robust method for discovering examples in the video data using Kanade-Lucas-Tomasi (KLT) feature tracks [8,9]. Building on the discovered examples, we introduce a novel formulation for unsupervised domain adaptation that adapts parameters of the detector from image

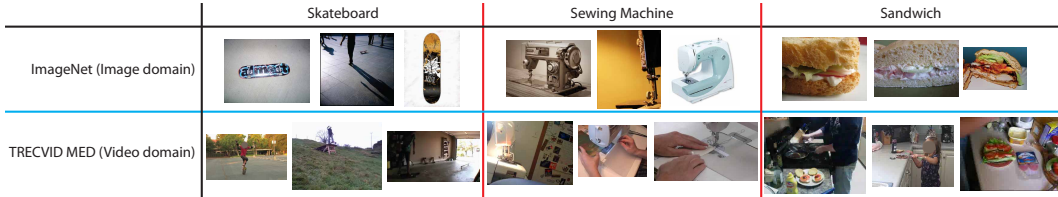

Figure 1: Images of the "Skateboard", "Sewing machine", and "Sandwich" classes taken from (top row) ImageNet [7] and (bottom row) TRECVID MED [1] illustrating differences in domain.

to video. This is done by iteratively including examples from the video data into the training set, while removing examples from the image data based on the *difficulty* of the examples. We define *easy* examples as ones with labels that can be predicted confidently (e.g., high likelihood, large distance from margin), and thus are more likely to be correct. In addition, it is common to have discriminative features that are only available in the target domain, which we term *target features*. For example, in the video domain, there are contextual features in the spatial and temporal vicinity of our detected object that we can take advantage of when performing detection. Our approach is able to incorporate the learning of parameters for these target features into a single objective.

## 2  Related Work

Most relevant are works that also deal with adapting detectors to video [10–13], but these works typically deal with a constrained set of videos and limited object classes. The work of [14] deals with a similar problem, but they adapt detectors from video to image. Our overall method is also similar to [15], in which we adopt an incremental approach to learn object category models.

Our setting is closely related to the domain adaptation problem, which has been studied extensively in vision settings. Several previous approaches focus on learning feature transformations between domains [16–18]. More similar to our method are approaches based on optimizing Support Vector Machine (SVM) related objectives [19–24] or joint cost functions [25], that treat the features as fixed and seek to adapt parameters of the classifier from source to target domain. However, with the exception of [18, 25], previous works deal with *supervised* or *semi-supervised* domain adaptation, which require labeled data in the target domain to generate associations between the source and target domains. In our setting, *unsupervised* domain adaptation, the target domain examples are unlabeled, and we must simultaneously discover and label examples in addition to learning parameters.

The objective we optimize to learn our detector draws inspiration from [26–28], in which we include and exclude the loss of certain examples using binary-valued indicator variables. Although our formulation is similar to [27, 28], our method is iterative and anneals weights that govern the number of examples to use, which is similar to the idea of self-paced learning [26], where a single weight is decreased to eventually include the loss of all examples in the objective. However, our method is different from [26] in that we have three sets of weights that govern the source examples, target examples, and target features. The weights are annealed in different directions, giving us the flexibility to iteratively include examples from the target domain, exclude examples from the source domain, and include parameters for the target features. In addition, our objective is able to incorporate target features, which is novel and not considered in [26–28].

Previous works have also considered ideas similar to our target features [29–32]. The work of [29] considers feature augmentation, but only with observed features common to both domains. Unobserved features in the context of clustering are investigated in [31], but in their setting all examples are assumed to have the same unobserved features. In [30, 32], features or modalities unseen in the training data are used to help in testing. However, both works assume there exists relationships between the seen and unseen features, whereas our target features are completely unrestricted.

## 3  Our Approach

We begin by providing an overview of our approach to adapting object detectors, as illustrated in Figure 2, and then elaborate on each of the steps. We assume that we are given a large amount of unlabeled video data with positive instances of our object class within some of these videos.

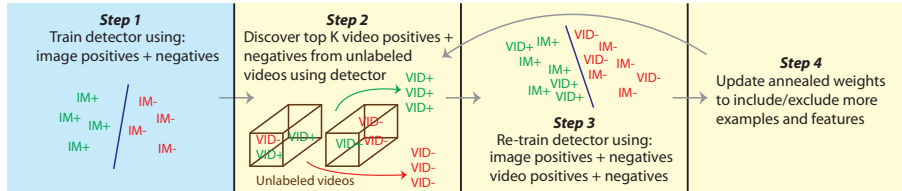

Figure 2: Overview of our algorithm. We start by initializing our detector using image positives and negatives (Step 1). We then proceed to enter a loop in which we discover the top $K$ video positives and negatives (Step 2), re-train our detector using these (Step 3), and then update the annealed parameters of the algorithm (Step 4).

We initialize our detector (Step 1 of Figure 2) by training a classifier on the labeled image positives and negatives, which we denote by our dataset $(\langle x_1, y_1 \rangle, ..., \langle x_n, y_n \rangle)$ with binary class labels $y_i \in \{-1, 1\}$. We consider a common method of learning weights $\boldsymbol{w}$ of a linear classifier:

$$\boldsymbol{w} = \arg\min_{\boldsymbol{w}} \left( r(\boldsymbol{w}) + C \sum_{i=1}^{n} Loss(x_i, y_i; \boldsymbol{w}) \right) \quad (1)$$

where $r(\cdot)$ is a regularizer over the weights, $Loss(\cdot)$ is a loss function over the training example, and $C$ controls the tradeoff between the two.

Our goal then is to discover the top $K$ positive and negative examples from the unlabeled videos, and to use these examples to help re-train our detector. We do not attempt to discover all instances, but simply a sufficient quantity to help adapt our detector to the video domain. To discover the top $K$ video positives and negatives (Step 2 of Figure 2), we utilize the strong prior of temporal continuity and score trajectory tracks instead of bounding boxes, which we describe in Section 3.1. Given the discovered examples, we optimize a novel objective inspired by self-paced learning [26] that simultaneously selects easy examples and trains a new detector (Step 3 of Figure 2). Using this new detector, we repeat this process of example discovery and detector training until convergence, as illustrated in Figure 2.

### 3.1 Discovering Examples in Video

In this step of the algorithm, we are given weights $\boldsymbol{w}$ of an object detector that can be used to score bounding boxes in video frames. A naive approach would run our detector on frames of video, taking the highest scoring and lowest scoring bounding boxes as the top $K$ video positives and negatives. Although reasonable, this method doesn't take advantage of temporal continuity in videos. An object that appears in one frame of a video is certain to appear close in neighboring frames as well. Previous works have shown this intuition to yield good results [10, 12, 13].

**Track-based scoring** Our key idea is to score trajectory tracks, rather than bounding boxes, as illustrated in Figure 3. We obtain tracks by running a KLT tracker on our videos, which tracks a sparse set of features over large periods of time. Because of the large number of unlabeled videos we have, we elect to extract KLT tracks rather than computing dense tracks using optical flow. Note that these tracks follow features, and so they may not correspond to centered locations of objects.

For each track, we consider the set of all bounding box placements $\mathcal{B}$ around it that intersect with the track. Each box placement $b_i \in \mathcal{B}$ is associated with a relative coordinate $(b_i^x, b_i^y)$ as well as a score $b_i^s$. The relative coordinate $(b_i^x, b_i^y)$ is the point within the box (relative to the top-left corner of the box) that intersects the track. Using this coordinate, we can compute the position of $b_i$ at every point in time along the track. Note that the number of bounding boxes in $\mathcal{B}$ is only dependent on the dimensions of the detector and the scales we search over. The score $b_i^s$ is computed by pooling scores of the bounding box along multiple points of the track in time. We use average pooling in our experiments to be robust to noisy scores. Finally, we associate the track with the bounding box $b_{\max}$ with the highest score, and use the score $b_{\max}^s$ as the score of the track.

After scoring each track in our unlabeled videos, we select the top and bottom few scoring tracks, and extract bounding boxes from each using the associated box coordinates $(b_{\max}^x, b_{\max}^y)$ to get our top $K$ video positives and negatives. The boxes are extracted by sampling frames along the track.

**Advantages** Compared to the naive approach without tracks, this approach allows us to recover from false detections with high scores, which are common for weak detectors, as it is less likely that

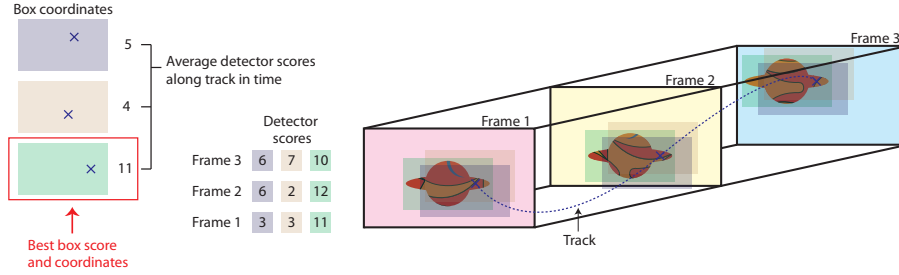

Figure 3: For a given KLT track, we consider all bounding box placements that intersect with it, denoted by the colored rectangular boxes. The purple cross denotes the intersection coordinates $(b_i^x, b_i^y)$ for each box. For each box, we average the scores at each point along the track, and take the box with the maximum score as the score and associated bounding box coordinates for this track.

there will be multiple false detections with high scores along a KLT track. Similarly, if the detection scores are consistently high along many points of a track, we can be more confident of the object's presence along the track. Hence, we can obtain novel examples of the object from various points of the track that had low scores, since we know the trajectory should correspond to the object. The same intuitions hold for true detections with low scores and obtaining negative examples.

## 3.2 Self-Paced Domain Adaptation

In this step of the algorithm, we are given the discovered top $K$ video positives and negatives, which we denote by the dataset $(\langle z_1, h_1 \rangle, ..., \langle z_k, h_k \rangle)$. Together with our original dataset $(\langle x_1, y_1 \rangle, ..., \langle x_n, y_n \rangle)$, we would like to learn a new detector.

A simple method would be to re-train our detector with both datasets using Equation 1. However, we typically aren't certain that the labels $\boldsymbol{h}$ are correct, especially in the first iteration when our detector is trained solely from the image examples. Ideally, we would like to re-train with a set of easier examples whose labels we are confident of first, and then re-discover video examples with this new detector. We would also like to stop learning from examples we are unsure of in the image domain, as they may be the examples most affected by the differences in domain. By repeating this process, we can avoid bad examples and iteratively refine our set of top $K$ video positives and negatives before having to train with all of them.

Formulating this intuition, our algorithm selects easier examples to learn from in the discovered video examples, and simultaneously selects harder examples in the image examples to stop learning from. An example is *difficult* if it has a large loss, as we are not confident in its correct label. The number of examples selected from the video examples and image examples are governed by weights that will be annealed over iterations (Step 4 of Figure 2).

**Basic approach** We start by introducing our approach without target features. We introduce binary variables $v_1, ..., v_n$ for the source domain (image) examples, and binary variables $u_1, ..., u_k$ for the target domain (video) examples. A value of 0 indicates that an example is difficult, and so we would like to remove its loss from consideration in the objective function. To prevent the algorithm from assigning all examples to be difficult, we introduce parameters $K^{source}$ and $K^{target}$ that control the number of examples considered from the source and target domain, respectively.

$$(\boldsymbol{w}_{t+1}, \boldsymbol{v}_{t+1}, \boldsymbol{u}_{t+1}) = \underset{\boldsymbol{w}, \boldsymbol{v}, \boldsymbol{u}}{\arg \min} \left( r(\boldsymbol{w}) + C \Big( \sum_{i=1}^{n} v_i Loss(x_i, y_i; \boldsymbol{w}) + \sum_{j=1}^{k} u_j Loss(z_j, h_j; \boldsymbol{w}) \Big) \right.$$
$$\left. - \frac{1}{K^{source}} \sum_{i=1}^{n} v_i - \frac{1}{K^{target}} \sum_{j=1}^{k} u_j \right) \qquad (2)$$

If $K^{target}$ is large, the algorithm prefers to consider only easy target examples with a small $Loss(\cdot)$, and the same is true for $K^{source}$. In the annealing of the weights for the algorithm (Step 4 of Figure 2), we decrease $K^{target}$ and increase $K^{source}$ to iteratively include more examples from the target domain and decrease examples from the source domain.

Similar to self-paced learning [26], we obtain a *tight* relaxation when allowing the binary variables $\boldsymbol{v}$ and $\boldsymbol{u}$ to take on any value in the interval $[0, 1]$. With the choice of $r(\cdot)$ and $Loss(\cdot)$ convex in $\boldsymbol{w}$, the problem becomes a bi-convex problem, and can be solved by alternating between (1) solving for $\boldsymbol{w}$ given $\boldsymbol{v}$ and $\boldsymbol{u}$, and (2) solving for $\boldsymbol{v}$ and $\boldsymbol{u}$ given $\boldsymbol{w}$. We refer the reader to [26] for further intuitions on the binary variables and annealed weights.

**Leveraging target features** Often, the target domain we are adapting to has additional features we can take advantage of. At the start, when we've only learned from a few examples in our target domain, we do not wish to rely on these rich and expressive features, as they can easily cause us to overfit. However, as we iteratively adapt to the target domain and build more confidence in our detector, we can start utilizing these target features to help with detection. The inclusion of these features is naturally self-paced as well, and can be easily integrated into our framework.

We assume there are a set of features that are shared between the source and target domains as $\phi_{shared}$, and a set of target domain-only features as $\phi_{target}$: $\phi = [\phi_{shared} \quad \phi_{target}]$. The weights $\boldsymbol{w}$ we want to learn can now be divided into $\boldsymbol{w}_{shared}$ and $\boldsymbol{w}_{target}$: $\boldsymbol{w} = [\boldsymbol{w}_{shared} \quad \boldsymbol{w}_{target}]$. Since the source data doesn't have $\phi_{target}$ features, we initialize those features to be 0 so that $\boldsymbol{w}_{target}$ doesn't affect the loss on the source data. The new objective function is formulated as:

$$(\boldsymbol{w}_{t+1}, \boldsymbol{v}_{t+1}, \boldsymbol{u}_{t+1}) = \underset{\boldsymbol{w}, \boldsymbol{v}, \boldsymbol{u}}{\arg\min} \Bigg( r(\boldsymbol{w}) + C\Big( \sum_{i=1}^{n} v_i Loss(x_i, y_i; \boldsymbol{w}) + \sum_{j=1}^{k} u_j Loss(z_j, h_j; \boldsymbol{w}) \Big)$$

$$+ \frac{1}{K^{feat}} ||\boldsymbol{w}_{target}||_1 - \frac{1}{K^{source}} \sum_{i=1}^{n} v_i - \frac{1}{K^{target}} \sum_{j=1}^{k} u_j \Bigg) \quad (3)$$

This is similar to Equation 2, with the addition of the $L_1$ norm term $\frac{1}{K^{feat}} ||\boldsymbol{w}_{target}||_1$. To anneal the weights for target features, we increase $K^{feat}$ to iteratively reduce the $L_1$ norm on the target features so that $\boldsymbol{w}_{target}$ can become non-zero. Intuitively, we are forcing the weights $\boldsymbol{w}$ to only use shared features first, and to consider more target features when we have a better model of the target domain. The optimization can be solved in the same manner as Equation 2. We can also approximate the $L_1$ norm term for all target features to be effectively binary, forcing $K^{feat}$ to be 0 initially and switching to $\infty$ at a particular iteration. This amounts to only considering target features after a certain iteration, and is done in our experiments for more tractable learning.

## 4 Experiments

We present experimental results for adapting object detectors on the 2011 TRECVID Multimedia Event Detection (MED) dataset [1] and LabelMe Video [2] dataset. For both, we select a set of objects which are known to appear in the videos. We used images from ImageNet [7] for the labeled image data, as there are a large number of diverse categories on ImageNet that correspond well with the objects that appear in the videos. We evaluate the detection performance of our models with the measure used in the PASCAL Visual Object Classes challenge [33], and report average precision (AP) scores for each class. The detection scores are computed on annotated video frames from the respective video datasets that are disjoint from the unlabeled videos used in the adapting stage.

### 4.1 Implementation Details

In our experiments, we use object detectors that are rectangular filters over Histogram-of-Gradient (HOG) features [34]. We use $L_2$ regularization for $r(\cdot)$ and hinge loss for $Loss(\cdot)$, which corresponds to the standard linear SVM formulation. For target features, we use contextual spatial features. The spatial features are taken to be HOG features bordering the object with dimensions half the size of the object bounding box. As described previously, we approximate the $L_1$ norm term to be binary to enable fast training using LIBLINEAR [35] when optimizing for $\boldsymbol{w}$. This also further decreases the number of model parameters needed to be searched over.

To isolate the effects of adaptation and better analyze our method, we restrict our experiments to the setting in which we fix the video negatives, and focus our problem on adapting from the labeled image positives to the unlabeled video positives. This scenario is realistic and commonly seen, as we can easily obtain video negatives by sampling from a set of unlabeled or weakly-labeled videos.

Table 1: Average Precision (AP) values for detection on the TRECVID MED dataset

| Object | InitialBL | VideoPosBL | Our method(nt) | Our method(full) | Gopalan *et al*. [18] (PLS) | Gopalan *et al*. [18] (SVM) |
|---|---|---|---|---|---|---|
| Skateboard | 4.29% | 2.89% | 10.44% | **10.44%** | 0.04% | 0.94% |
| Animal | 0.41% | 0.40% | 0.39% | **3.76%** | 0.16% | 0.24% |
| Tire | 11.22% | 11.04% | 15.54% | **15.54%** | 0.60% | 15.52% |
| Vehicle | 4.03% | **4.08%** | 3.57% | 3.57% | 3.33% | 3.16% |
| Sandwich | 10.07% | 9.85% | 9.45% | **12.49%** | 0.21% | 6.68% |
| Sewing machine | 9.76% | 9.71% | 10.35% | **10.35%** | 0.12% | 3.81% |
| Mean AP | 6.63% | 6.33% | 8.29% | **9.36%** | 0.74% | 5.06% |

Table 2: Average Precision (AP) values for detection on the LabelMe Video dataset

| Object | InitialBL | VideoPosBL | Our method(nt) | Our method(full) | Gopalan *et al*. [18] (PLS) | Gopalan *et al*. [18] (SVM) |
|---|---|---|---|---|---|---|
| Car | 2.60% | 2.13% | 2.15% | **9.18%** | 0.34% | 1.00% |
| Boat | 0.22% | 0.22% | 0.22% | 0.22% | 0.05% | **0.32%** |
| Bicycle | 19.85% | 19.76% | 20.27% | **20.27%** | 0.21% | 16.32% |
| Dog | 1.74% | 2.42% | 2.47% | **4.75%** | 0.18% | 1.48% |
| Keyboard | 0.41% | **0.67%** | 0.59% | 0.59% | 0.13% | 0.09% |
| Mean AP | 4.96% | 5.04% | 5.14% | **7.00%** | 0.18% | 3.84% |

**Model parameters** In our experiments, we fix the total number of iterations to 5 for tractable training time. For the $K^{target}$ and $K^{source}$ weights, we set values for the first and final iterations, and linearly interpolate values for the remaining iterations in between. For the $K^{target}$ weight, we estimate the weights so that we start by considering only the video examples that have no loss, and end with all video examples considered. For the $K^{source}$ weight, we vary the ending weight so that differing numbers of source examples are left for training at the final iteration. For the target features, we set the algorithm to allow target features at the midpoint of total iterations. Based on the number of KLT tracks extracted, we set the top $K$ examples to be between 100 and 500.

**Model selection** The free model parameters that can be varied are the number of top $K$ examples to discover, the ending $K^{source}$ weight, and whether or not to use target features. In our results, we perform model selection by comparing the distribution of scores on the discovered video positives. The distributions are compared between the initial models from iteration 1 for different model parameters to select $K$ and $K^{source}$, and between the final iteration 5 models for different model parameters to determine the use of target features. This allows us to evaluate the strength of the initial model trained on the image positives and video negatives, as well as our final adapted model. We select the model with the distributions indicating the highest confidence in its classification boundary.

## 4.2 Baseline Comparisons

**InitialBL** This baseline is the intial detector trained only on image positives and video negatives.

**VideoPosBL** This baseline uses the intial detector to discover the top $K$ video positives from the unlabeled video, then trains with all these examples without iterating. Thus, it incorporates our idea of discovering video positives by scoring tracks and re-training, but does not use self-paced domain adaptation for learning weights. It can also be thought of as our method run for one iteration.

**Our method(nt)** This baseline uses our full method with the exception of target features.

**Gopalan *et al*.** This is a state-of-the-art method for unsupervised domain adaptation [18] that models the domain shift in feature space. Since we are not given labels in the target domain, most previous methods for domain adaptation cannot be applied to our setting. This method samples subspaces along the geodesic between the source and target domains on the Grassman manifold. Using projections of both source and target data onto the common subspaces, they learn a discriminative classifier using partial least squares (PLS) with available labels from either domains. We ran their code using their suggested parameter settings to obtain results for their method on our task. We also show results for their method using a linear SVM as the classifier to allow for fair comparisons.

## 4.3 TRECVID MED

The 2011 TRECVID MED dataset [1] consists of a collection of Internet videos collected by the Linguistic Data Consortium from various Internet video hosting sites. There are a total of 15 complex events, and videos are labeled with either an event class or no label, where an absence of label indicates the video belongs to no event class. We select 6 object classes to learn object detectors for because they are commonly present in selected events: "Skateboard", "Animal", "Tire", "Vehicle", "Sandwich", and "Sewing machine". These objects appear respectively in the events "Attempting a

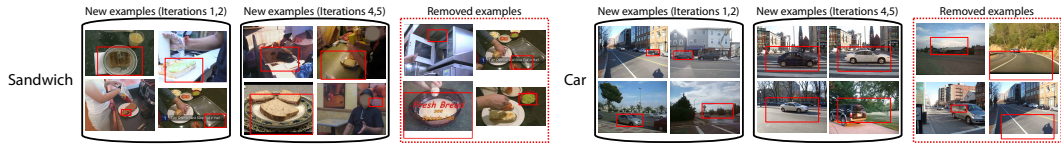

Figure 4: Discovered top $K$ video positives using our method for "Sandwich" and "Car". After sets of iterations, we show samples of newly discovered video positives (red boxes) that were not in the set of top $K$ of previous iterations (left, middle columns). We also show bad examples that were removed from the top $K$ over all iterations (right column). As our model adapts, it is able to iteratively refine its set of top $K$ video positives. Figure best viewed magnified and in color.

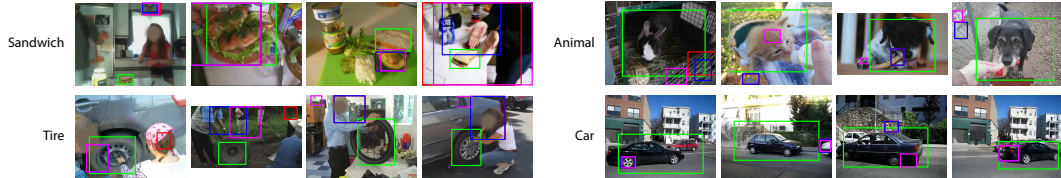

Figure 5: Detections for "Sandwich", "Tire", "Animal", and "Car". Green boxes detections from our method, red boxes detections from "InitialBL", blue boxes detections from "VideoPosBL", and magenta boxes detections from Gopalan *et al.*(SVM). Figure best viewed magnified and in color.

board trick", "Feeding an animal", "Changing a vehicle tire", "Getting a vehicle unstuck", "Making a sandwich", and "Working on a sewing project". The video negatives were randomly sampled from the videos that were labeled with no event class.

To test our algorithm, we manually annotated approximately 200 frames with bounding boxes of positive examples for each object, resulting in 1234 annotated frames total from over 500 videos, giving us a diverse set of situations the objects can appear in. For each object, we use 20 videos from the associated event as unlabeled video training data. Results are given in Table 1.

### 4.4   LabelMe Video

LabelMe Video [2] is a database of real-world videos that contains a large set of annotations including object category, shape, motion, and activity information. We use the database of videos that was introduced in the original paper [2]. There are a large number of objects that are annotated in this database, and we select the most frequently occuring objects that are not scene parts, resulting in 5 objects: "Car", "Boat", "Bicycle", "Dog", and "Keyboard". The video negatives were randomly sampled from the videos that were not annotated with any of these objects.

We extract more than 200 frames with positive examples for each object class, resulting in a test set of 1137 images. For each object class, we use the remaining videos that contain the object as the unlabeled video training data, resulting in around 9 videos per object. Results are given in Table 2.

## 5   Discussion

From our results in Tables 1 and 2, we can observe similar patterns for most object classes. First, we note that the "VideoPosBL" baseline typically performs on par with the "InitialBL" baseline, and rarely does it post a slight gain in performance. This shows that if we discover the top $K$ video positives and re-train our detector with all of them, we do not obtain consistent gains in performance. Our method of self-paced domain adaptation is crucial in this case, as we can see that our full method typically outperforms all other methods by significant margins. As illustrated in Figure 4, our method is able to add new video positives from iteration to iteration that are good examples, and remove bad examples at the same time. The method of Gopalan *et al.* [18] performs very poorly when used in conjunction with the PLS classifier, but becomes more competitive when used with an SVM. However, even then their method performs much worse than our method for nearly all object classes, as it is difficult to model the underlying domain shift in feature space. This also serves to illustrate the difficulty of our problem, as poor adaptation can lead to results worse than the baselines. We show visualizations of our detections compared to baseline methods in Figure 5.

Observing the visualizations of the learned weights for the "Tire", "Car" and "Sandwich" classes in Figure 6, we see that weights trained with our method exhibit more clearly defined structure

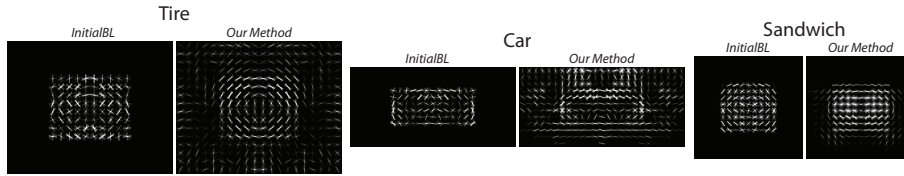

Figure 6: Visualizations of the positive HOG weights learned for three classes for the "InitialBL" baseline and our method. The spatial context weights are 0 for "InitialBL" because it does not consider target features, resulting in a black border. Figure best viewed magnified and in color.

than the "InitialBL" baseline. The target features also help performance significantly. By capturing interesting patterns in the spatial context, difficult objects can become easier to detect in the target domain. For the "Sandwich" class, we can see circular weights in the spatial context surrounding the sandwich, suggesting that sandwiches typically appear on plates, and for "Car", we can clearly distinguish weights for the road beneath the car object. We observe an average AP gain of 3.93% for classes that choose models with target features versus no target features. Note that we chose to use simple spatial context as target features in our models, as they are fast to implement and easily incorporated. However, we hypothesize that the inclusion of more complex target features such as temporal movement could help our method achieve even better results.

We observe that for the "Vehicle" and "Keyboard" classes, the "VideoPosBL" baseline performs better than our full method. Although this is not a common occurrence, it can happen when our method of self-paced domain adaptation replaces good video positives taken in the first iteration with bad examples in future iterations. This situation arises when there are incorrect examples present in the easiest of the top $K$ video positives, causing our detector to re-train and iteratively become worse. If we had better methods for model selection, we could also search over the number of total iterations as a model parameter, which would include the "VideoPosBL" model in our set of models to select over, as it is essentially our method run for a single iteration.

## 6  Conclusion

In this paper we have introduced an approach for adapting detectors from image to video. To discover examples in the unlabeled video data, we classify tracks instead of bounding boxes, allowing us to leverage temporal continuity to avoid spurious detections, and to discover examples we would've otherwise missed. Furthermore, we introduce a novel self-paced domain adaptation algorithm that allows our detector to iteratively adapt from source to target domain, while also considering target features unique to the target domain. Our formulation is general, and can be applied to various other problems in domain adaptation. We've shown convincing results that illustrate the benefit of our approach to adapting object detectors to video.

Possible directions for future work could include better methods for model selection. A measure that would allow us to estimate our performance on the target domain with theoretical guarantees would be an interesting direction. Another possible direction would be to relax the assumption of having no labeled target domain examples, and to formulate similar methods for this scenario.

**Acknowledgments.** We thank Tianshi Gao for helpful discussions. We also thank Chris Baldassano, Eric Huang, Jia Deng, and Olga Russakovsky for helpful comments on the paper. This work was supported by the Defense Advanced Research Projects Agency under Contract No. HR0011-08-C-0135 and by the Intelligence Advanced Research Projects Activity (IARPA) via Department of Interior National Business Center contract number D11PC20069. The U.S. Government is authorized to reproduce and distribute reprints for Governmental purposes notwithstanding any copyright annotation thereon. Disclaimer: The views and conclusions contained herein are those of the authors and should not be interpreted as necessarily representing the official policies or endorsements, either expressed or implied, of DARPA, IARPA, DoI/NBC, or the U.S. Government.

## References

[1] P. Over, G. Awad, M. Michel, J. Fiscus, W. Kraaij, and A. F. Smeaton. Trecvid 2011 – an overview of the goals, tasks, data, evaluation mechanisms and metrics. In *TRECVID 2011*. NIST, USA, 2011.

[2] J. Yuen, B. C. Russell, C. Liu, and A. Torralba. Labelme video: Building a video database with human annotations. In *ICCV*, 2009.

[3] C. Schuldt, I. Laptev, and B. Caputo. Recognizing human actions: A local svm approach. In *ICPR*, 2004.

[4] L. Gorelick, M. Blank, E. Shechtman, M. Irani, and R. Basri. Actions as space-time shapes. *IEEE TPAMI*, 2007.

[5] J. C. Niebles, C.-W. Chen, and L. Fei-Fei. Modeling temporal structure of decomposable motion segments for activity classification. In *ECCV*, 2010.

[6] K. Tang, L. Fei-Fei, and D. Koller. Learning latent temporal structure for complex event detection. In *CVPR*, 2012.

[7] J. Deng, W. Dong, R. Socher, L.-J. Li, K. Li, and L. Fei-Fei. ImageNet: A Large-Scale Hierarchical Image Database. In *CVPR*, 2009.

[8] B. D. Lucas and T. Kanade. An iterative image registration technique with an application to stereo vision. In *IJCAI*, 1981.

[9] C. Tomasi and T. Kanade. Detection and tracking of point features. Technical report, CMU, 1991.

[10] P. Sharma, C. Huang, and R. Nevatia. Unsupervised incremental learning for improved object detection in a video. In *CVPR*, 2012.

[11] X. Wang, G. Hua, and T. X. Han. Detection by detections: Non-parametric detector adaptation for a video. In *CVPR*, 2012.

[12] M. Yang, S. Zhu, F. Lv, and K. Yu. Correspondence driven adaptation for human profile recognition. In *CVPR*, 2011.

[13] N. Cherniavsky, I. Laptev, J. Sivic, and A. Zisserman. Semi-supervised learning of facial attributes in video. In *ECCV 2010*, 2010.

[14] A. Prest, C. Leistner, J. Civera, C. Schmid, and V. Ferrari. Learning object class detectors from weakly annotated video. In *CVPR*, 2012.

[15] L.-J. Li and L. Fei-Fei. OPTIMOL: automatic Online Picture collecTion via Incremental MOdel Learning. *IJCV*, 2009.

[16] K. Saenko, B. Kulis, M. Fritz, and T. Darrell. Adapting visual category models to new domains. In *ECCV*, 2010.

[17] B. Kulis, K. Saenko, and T. Darrell. What you saw is not what you get: Domain adaptation using asymmetric kernel transforms. In *CVPR*, 2011.

[18] R. Gopalan, R. Li, and R. Chellappa. Domain adaptation for object recognition: An unsupervised approach. In *ICCV*, 2011.

[19] A. Bergamo and L. Torresani. Exploiting weakly-labeled web images to improve object classification: a domain adaptation approach. In *NIPS*, 2010.

[20] G. Schweikert, C. Widmer, B. Schölkopf, and G. Rätsch. An empirical analysis of domain adaptation algorithms for genomic sequence analysis. In *NIPS*, 2008.

[21] J. Yang, R. Yan, and A. G. Hauptmann. Cross-domain video concept detection using adaptive svms. In *ACM Multimedia*, 2007.

[22] L. Duan, D. Xu, I. W.-H. Tsang, and J. Luo. Visual event recognition in videos by learning from web data. In *CVPR*, 2010.

[23] T. Joachims. Transductive inference for text classification using support vector machines. In *ICML*, 1999.

[24] T. Tommasi, F. Orabona, and B. Caputo. Safety in numbers: Learning categories from few examples with multi model knowledge transfer. In *CVPR*, 2010.

[25] C. Zhang, R. Hamid, and Z. Zhang. Taylor expansion based classifier adaptation: Application to person detection. In *CVPR*, 2008.

[26] P. Kumar, B. Packer, and D. Koller. Self-paced learning for latent variable models. In *NIPS*, 2010.

[27] J. J. Lim, R. Salakhutdinov, and A. Torralba. Transfer learning by borrowing examples for multiclass object detection. In *NIPS*, 2011.

[28] T. Gao and D. Koller. Discriminative learning of relaxed hierarchy for large-scale visual recognition. In *ICCV*, 2011.

[29] H. D. III. Frustratingly easy domain adaptation. In *ACL*, 2007.

[30] B. Taskar, M.-F. Wong, and D. Koller. Learning on the test data: Leveraging 'unseen' features. In *ICML*, 2003.

[31] E. Krupka and N. Tishby. Generalization in clustering with unobserved features. In *NIPS*, 2005.

[32] C. M. Christoudias, R. Urtasun, M. Salzmann, and T. Darrell. Learning to recognize objects from unseen modalities. In *ECCV*, 2010.

[33] M. Everingham, L. Van Gool, C. K. I. Williams, J. Winn, and A. Zisserman. The pascal visual object classes (voc) challenge. *IJCV*, 2010.

[34] N. Dalal and B. Triggs. Histograms of oriented gradients for human detection. In *CVPR*, 2005.

[35] R.-E. Fan, K.-W. Chang, C.-J. Hsieh, X.-R. Wang, and C.-J. Lin. LIBLINEAR: A library for large linear classification. *JMLR*, 2008.

